# Constructive Learning Using Internal Representation Conflicts

**Laurens R. Leerink and Marwan A. Jabri**
Systems Engineering & Design Automation Laboratory
Department of Electrical Engineering
The University of Sydney
Sydney, NSW 2006, Australia

## Abstract

We present an algorithm for the training of feedforward and recurrent neural networks. It detects internal representation conflicts and uses these conflicts in a constructive manner to add new neurons to the network. The advantages are twofold: (1) starting with a small network neurons are only allocated when required; (2) by detecting and resolving internal conflicts at an early stage learning time is reduced. Empirical results on two real-world problems substantiate the faster learning speed; when applied to the training of a recurrent network on a well researched sequence recognition task (the Reber grammar), training times are significantly less than previously reported.

## 1 Introduction

Selecting the optimal network architecture for a specific application is a nontrivial task, and several algorithms have been proposed to automate this process. The first class of network adaptation algorithms start out with a redundant architecture and proceed by pruning away seemingly unimportant weights (Sietsma and Dow, 1988; Le Cun et al, 1990). A second class of algorithms starts off with a sparse architecture and grows the network to the complexity required by the problem. Several algorithms have been proposed for growing feedforward networks. The upstart algorithm of Frean (1990) and the cascade-correlation algorithm of Fahlman (1990) are examples of this approach.

The cascade correlation algorithm has also been extended to recurrent networks (Fahlman, 1991), and has been shown to produce good results. The recurrent cascade-correlation (RCC) algorithm adds a fully connected layer to the network after every step, in the process attempting to correlate the output of the additional layer with the error. In contrast, our proposed algorithm uses the statistical properties of the weight adjustments produced during batch learning to add additional units.

The RCC algorithm will be used as a baseline against which the performance of our method will be compared. In a recent paper, Chen et al (1993) presented an algorithm which adds one recurrent neuron with small weights every $N$ epochs. However, no significant improvement in training speed was reported over training the corresponding fixed size network, and the algorithm will not be further analyzed. To the authors knowledge little work besides the two mentioned papers have applied constructive algorithms to recurrent networks.

In the majority of our empirical studies we have used partially recurrent neural networks, and in this paper we will focus our attention on such networks. The motivation for the development of this algorithm partly stemmed from the long training times experienced with the problems of phoneme and word recognition from continuous speech. However, the algorithm is directly applicable to feedforward networks. The same criteria and method used to add recurrent neurons to a recurrent network can be used for adding neurons to any hidden layer of a feed-forward network.

## 2   Architecture

In a standard feedforward network, the outputs only depend on the current inputs, the network architecture and the weights in the network. However, because of the temporal nature of several applications, in particular speech recognition, it might be necessary for the network to have a short term memory.

Partially recurrent networks, often referred to as Jordan (1989) or Elman (1990) networks, are well suited to these problems. The architecture examined in this paper is based on the work done by Robinson and Fallside (1991) who have applied their recurrent error propagation network to continuous speech recognition.

A common feature of all partially recurrent networks is that there is a special set of neurons called context units which receive feedback signals from a previous time step. Let the values of the context units at time $t$ be represented by $C(t)$. During normal operation the input vector at time $t$ are applied to the input nodes $I(t)$, and during the feedforward calculation values are produced at both the output nodes $O(t + 1)$ and the context units $C(t + 1)$. The values of the context units are then copied back to the input layer for use as input in the following time step.

Several training algorithms exist for training partially recurrent neural networks, but for tasks with large training sets the back-propagation through time (Werbos, 1990) is often used. This method is computationally efficient and does not use any approximations in following the gradient. For an application where the time information is spread over $T$. input patterns, the algorithm simply duplicates the network $T$ times - which results in a feedforward network that can be trained by a variation of the standard backpropagation algorithm.

# 3   The Algorithm

For partially recurrent networks consisting of input, output and context neurons, the following assertions can be made:

- The role of the context units in the network is to extract and store all relevant prior information from the sequence pertaining to the classification problem.

- For weights entering context units the weight update values accumulated during batch learning will eventually determine what context information is stored in the unit (the sum of the weight update values is larger than the initial random weights).

- We assume that initially the number of context units in the network is insufficient to implement this extraction and storage of information (we start training with a small network). Then, at different moments in time during the recognition of long temporal sequences, a context unit could be required to preserve several different contexts.

- These conflicts are manifested as distinct peaks in the distribution of the weight update values during the epoch.

All but the last fact follows directly from the network architecture and requires no further elaboration. The peaks in the distribution of the weight update values are a result of the training algorithm attempting to adjust the value of the context units in order to provide a context value that will resolve short-term memory requirements.

After the algorithm had been developed, it was discovered that this aspect of the weight update values had been used in the past by Wynne-Jones (1992) and in the *Meiosis Networks* of Hanson (1990). The method of Wynne-Jones (1992) in particular is very closely related; in this case principal component analysis of the weight updates and the Hessian matrix is used to detect oscillating nodes in fully trained feed-forward networks. This aspect of backpropagation training is fully discussed in Wynne-Jones (1992), to which reader is referred for further details.

The above assertions lead to the proposed training algorithm, which states that if there are distinct maxima in the distribution of weight update values of the weights entering a context unit, then this is an indication that the batch learning algorithm requires this context unit for the storage of more than one context.

If this conflict can be resolved, the network can effectively store all the contexts required, leading to a reduction in training time and potentially an increase in performance.

The training algorithm is given below (the mode of the distribution is defined as the number of distinct maxima):

```
For all context units {
  Set N = modality of the distribution of weight update values;
  If N > 1 then {
    Add N-1 new context units to the network which are identical
    (in terms of weighted inputs) to the current context unit.
```

```
    Adjust each of these N context units (including the
    original) by the weight update value determined by each
    maxima (the average value of the mode).

    Adjust all weights leaving these N context units so that the
    addition of the new units do not affect any subsequent layers
    (division by N). This ensures that the network retains all
    previously acquired knowledge.
  }
}
```

The main problem in the implementation of the above algorithm is the automatic detection of significant maxima in the distribution of weight updates. A standard statistical approach for the determination of the modality (the number of maxima) of a distribution of noisy data is to fit a curve of a certain predetermined order to the data. The maxima (and minima) are then found by setting the derivative to zero. This method was found to be unsuitable mainly because after curve fitting it was difficult to determine the significance of the detected peaks.

It was decided that only instances of bi-modality and tri-modality were to be identified, each corresponding to the addition of one or two context units. The following heuristic was constructed:

- Calculate the mean and standard deviation of the weight update values.
- Obtain the maximum value in the distribution.
- If there are any peaks larger than 60% of the maxima outside one standard deviation of the mean, regard this as significant.

This heuristic provided adequate identification of the modalities. The distribution was divided into three areas using the mean ± the standard deviation as boundaries. Depending on the number of maxima detected, the average within each area is used to adjust the weights.

## 4   Discussion

According to our algorithm it follows that if at least one weight entering a context unit has a multi-modal distribution, then that context unit is duplicated. In the case where multi-modality is detected in more than one weight, context units were added according to the highest modality.

Although this algorithm increases the computational load during training, the standard deviation of the weight updates rapidly decreases as the network converges. The narrowing of the distribution makes it more difficult to determine the modality. In practice it was only found useful to apply the algorithm during the initial training epochs, typically during the first 20.

During simulations in which strong multi-modalities were detected in certain nodes, frequently the multi-modalities would persist in the newly created nodes. In this

manner a strong bi-modality would cause one node to split into two, the two nodes to grow to four, etc. This behaviour was prevented by disabling the splitting of a node for a variable number of epochs after a multi-modality had been detected. Disabling this behaviour for two epochs provided good results.

## 5  Simulation Results

The algorithm was evaluated empirically on two different tasks:

- Phoneme recognition from continuous multi-speaker speech using the TIMIT (Garofolo, 1988) acoustic-phonetic database.
- Sequence Recognition: Learning a finite-state grammar from examples of valid sequences.

For the phoneme recognition task the algorithm decreased training times by a factor of 2 to 10, depending on the size of the network and the size of the training set.

The sequence recognition task has been studied by other researchers in the past, notably Fahlman (1991). Fahlman compared the performance of the recurrent cascade correlation (RCC) network with that of previous results by Cleeremans et al (1989) who used an Elman (1990) network. It was concluded that the RCC algorithm provides the same or better performance than the Elman network with less training cycles on a smaller training set. Our simulations have shown that the recurrent error propagation network of Robinson and Fallside (1991), when trained with our constructive algorithm and a learning rate adaptation heuristic, can provide the same performance as the RCC architecture in 40% fewer training epochs using a training set of the same size. The resulting network has the same number of weights as the minimum size RCC network which correctly solves this problem.

Constructive algorithms are often criticized in terms of efficiency, i.e. "Is the increase in learning speed due to the algorithm or just the additional degrees of freedom resulting from the added neuron and associated weights?". To address this question several simulations were conducted on the speech recognition task, comparing the performance and learning time of a network with $N$ fixed context units to that of a network with small number of context units and growing a network with a maximum of $N$ context units. Results indicate that the constructive algorithm consistently trains faster, even though both networks often have the same final performance.

## 6  Summary

In this paper the statistical properties of the weight update values obtained during the training of a simple recurrent network using back-propagation through time have been examined. An algorithm has been presented for using these properties to detect internal representation conflicts during training and to use this information to add recurrent units to the network. Simulation results show that the algorithm decreases training time compared to networks which have a fixed number of context units. The algorithm has not been applied to feedforward networks, but can in principle be added to all training algorithms that operate in batch mode.

## References

Chen, D., Giles, C.L., Sun, G.Z., Chen, H.H., Lee, Y.C., Goudreau, M.W. (1993). Constructive Learning of Recurrent Neural Networks. In *1993 IEEE International Conference on Neural Networks,* **III**:1196-1201. Piscataway, NJ: IEEE Press.

Cleeremans, A., Servan-Schreiber, D., and McClelland, J.L. (1989). Finite State Automata and Simple Recurrent Networks. *Neural Computation* **1**:372-381.

Elman, J.L. (1990). Finding Structure in Time. *Cognitive Science* **14**:179-211.

Fahlman, S.E. and C. Lebiere (1990). The Cascade Correlation Learning Architecture. In D. S. Touretzky (ed.), *Advances in Neural Information Processing Systems 2,* 524-532. San Mateo, CA: Morgan Kaufmann.

Fahlman, S.E. (1991). The Recurrent Cascade Correlation Architecture. Technical Report CMU-CS-91-100. School of Computer Science, Carnegie Mellon University.

Frean, M. (1990). The Upstart Algorithm: A Method for Constructing and Training Feedforward Neural Networks. *Neural Computation* **2**:198-209.

Garofolo, J.S. (1988). Getting Started with the DARPA TIMIT CD-ROM: an Acoustic Phonetic Continuous Speech Database. National Institute of Standards and Technology (NIST), Gaithersburgh, Maryland.

Hanson, S.J. (1990). Meiosis Networks. In D. S. Touretzky (ed.), *Advances in Neural Information Processing Systems 2,* 533-541, San Mateo, CA: Morgan Kaufmann.

Jordan, M.I. (1989). Serial Order: A Parallel, Distributed Processing Approach. In *Advances in Connectionist Theory: Speech,* eds. J.L. Elman and D.E. Rumelhart. Hillsdale: Erlbaum.

Le Cun, Y., J.S. Denker, and S.A Solla (1990). Optimal Brain Damage. In D. S. Touretzky (ed.), *Advances in Neural Information Processing Systems 2,* 598-605. San Mateo, CA: Morgan Kaufmann.

Reber, A.S. (1967). Implicit learning of artificial grammars. *Journal of Verbal Learning and Verbal Behavior* **5**:855-863.

Robinson, A.J. and Fallside F. (1991). An error propagation network speech recognition system. *Computer Speech and Language* **5**:259-274.

Sietsma, J. and RJ.F Dow (1988). Neural Net Pruning-Why and How. In *IEEE International Conference on Neural Networks.* (San Diego 1988), **1**:325-333.

Wynne-Jones, M. (1992) Node Splitting: A Constructive Algorithm for Feed-Forward Neural Networks. In D. S. Touretzky (ed.), *Advances in Neural Information Processing Systems 4,* 1072-1079. San Mateo, CA: Morgan Kaufmann.

Werbos, P.J. (1990). Backpropagation Through Time, How It Works and How to Do It. *Proceedings of the IEEE,* **78**:1550-1560.